# Efficient Large-Scale Distributed Training of Conditional Maximum Entropy Models

**Gideon Mann**
Google
gmann@google.com

**Ryan McDonald**
Google
ryanmcd@google.com

**Mehryar Mohri**
Courant Institute and Google
mohri@cims.nyu.edu

**Nathan Silberman**
Google
nsilberman@google.com

**Daniel D. Walker**[*]
NLP Lab, Brigham Young University
danl4@cs.byu.edu

## Abstract

Training conditional maximum entropy models on massive data sets requires significant computational resources. We examine three common distributed training methods for conditional maxent: a distributed gradient computation method, a majority vote method, and a mixture weight method. We analyze and compare the CPU and network time complexity of each of these methods and present a theoretical analysis of conditional maxent models, including a study of the convergence of the mixture weight method, the most resource-efficient technique. We also report the results of large-scale experiments comparing these three methods which demonstrate the benefits of the mixture weight method: this method consumes less resources, while achieving a performance comparable to that of standard approaches.

## 1 Introduction

Conditional maximum entropy models [1, 3], conditional maxent models for short, also known as multinomial logistic regression models, are widely used in applications, most prominently for multi-class classification problems with a large number of classes in natural language processing [1, 3] and computer vision [12] over the last decade or more.

These models are based on the maximum entropy principle of Jaynes [11], which consists of selecting among the models approximately consistent with the constraints, the one with the greatest entropy. They benefit from a theoretical foundation similar to that of standard maxent probabilistic models used for density estimation [8]. In particular, a duality theorem for conditional maxent model shows that these models belong to the exponential family. As shown by Lebanon and Lafferty [13], in the case of two classes, these models are also closely related to AdaBoost, which can be viewed as solving precisely the same optimization problem with the same constraints, modulo a normalization constraint needed in the conditional maxent case to derive probability distributions.

While the theoretical foundation of conditional maxent models makes them attractive, the computational cost of their optimization problem is often prohibitive for data sets of several million points. A number of algorithms have been described for batch training of conditional maxent models using a single processor. These include generalized iterative scaling [7], improved iterative scaling [8], gradient descent, conjugate gradient methods, and second-order methods [15, 18].

This paper examines distributed methods for training conditional maxent models that can scale to very large samples of up to 1B instances. Both batch algorithms and on-line training algorithms such

---

[*]This work was conducted while at Google Research, New York.

as that of [5] or stochastic gradient descent [21] can benefit from parallelization, but we concentrate here on batch distributed methods.

We examine three common distributed training methods: a distributed gradient computation method [4], a majority vote method, and a mixture weight method. We analyze and compare the CPU and network time complexity of each of these methods (Section 2) and present a theoretical analysis of conditional maxent models (Section 3), including a study of the convergence of the mixture weight method, the most resource-efficient technique. We also report the results of large-scale experiments comparing these three methods which demonstrate the benefits of the mixture weight method (Section 4): this method consumes less resources, while achieving a performance comparable to that of standard approaches such as the distributed gradient computation method.[1]

## 2  Distributed Training of Conditional Maxent Models

In this section, we first briefly describe the optimization problem for conditional maximum entropy models, then discuss three common methods for distributed training of these models and compare their CPU and network time complexity.

### 2.1  Conditional Maxent Optimization problem

Let $\mathcal{X}$ be the input space, $\mathcal{Y}$ the output space, and $\Phi\colon \mathcal{X}\times\mathcal{Y}\to H$ a (feature) mapping to a Hilbert space $H$, which in many practical settings coincides with $\mathbb{R}^N$, $N=\dim(H)<\infty$. We denote by $\|\cdot\|$ the norm induced by the inner product associated to $H$.

Let $S=((x_1,y_1),\ldots,(x_m,y_m))$ be a training sample of $m$ pairs in $\mathcal{X}\times\mathcal{Y}$. A conditional maximum entropy model is a conditional probability of the form $p_{\mathbf{w}}[y|x]=\frac{1}{Z(x)}\exp(\mathbf{w}\cdot\Phi(x,y))$ with $Z(x)=\sum_{y\in Y}\exp(\mathbf{w}\cdot\Phi(x,y))$, where the weight or parameter vector $\mathbf{w}\in H$ is the solution of the following optimization problem:

$$\mathbf{w}=\operatorname*{argmin}_{\mathbf{w}\in H} F_S(\mathbf{w}) = \operatorname*{argmin}_{\mathbf{w}\in H} \lambda\|\mathbf{w}\|^2 - \frac{1}{m}\sum_{i=1}^{m}\log p_{\mathbf{w}}[y_i|x_i]. \qquad (1)$$

Here, $\lambda\geq 0$ is a regularization parameter typically selected via cross-validation. The optimization problem just described corresponds to an $L_2$ regularization. Many other types of regularization have been considered for the same problem in the literature, in particular $L_1$ regularization or regularizations based on other norms. This paper will focus on conditional maximum entropy models with $L_2$ regularization.

These models have been extensively used and studied in natural language processing [1, 3] and other areas where they are typically used for classification. Given the weight vector $\mathbf{w}$, the output $y$ predicted by the model for an input $x$ is:

$$y=\operatorname*{argmax}_{y\in\mathcal{Y}} p_{\mathbf{w}}[y|x] = \operatorname*{argmax}_{y\in\mathcal{Y}} \mathbf{w}\cdot\Phi(x,y). \qquad (2)$$

Since the function $F_S$ is convex and differentiable, gradient-based methods can be used to find a global minimizer $\mathbf{w}$ of $F_S$. Standard training methods such as iterative scaling, gradient descent, conjugate gradient, and limited-memory quasi-Newton all have the general form of Figure 1, where the update function $\Gamma\colon H\to H$ for the gradient $\nabla F_S(\mathbf{w})$ depends on the optimization method selected. $T$ is the number of iterations needed for the algorithm to converge to a global minimum. In practice, convergence occurs when $F_S(\mathbf{w})$ differs by less than a constant $\epsilon$ in successive iterations of the loop.

### 2.2  Distributed Gradient Computation Method

Since the points are sampled i.i.d., the gradient computation in step 3 of Figure 1 can be distributed across $p$ machines. Consider a sample $S=(S_1,\ldots,S_p)$ of $pm$ points formed by $p$ subsamples of

$$
\begin{aligned}
&1 \quad \mathbf{w} \leftarrow \mathbf{0} \\
&2 \quad \textbf{for } t \leftarrow 1 \textbf{ to } T \textbf{ do} \\
&3 \qquad \nabla F_S(\mathbf{w}) \leftarrow \textsc{Gradient}(F_S(\mathbf{w})) \\
&4 \qquad \mathbf{w} \leftarrow \mathbf{w} + \Gamma(\nabla F_S(\mathbf{w})) \\
&5 \quad \textbf{return } \mathbf{w}
\end{aligned}
$$

Figure 1: Standard Training

$$
\begin{aligned}
&1 \quad \mathbf{w} \leftarrow \mathbf{0} \\
&2 \quad \textbf{for } t \leftarrow 1 \textbf{ to } T \textbf{ do} \\
&3 \qquad \nabla F_S(\mathbf{w}) \leftarrow \textsc{DistGradient}(F_{S_k}(\mathbf{w}) \parallel p \text{ machines}) \\
&4 \qquad \mathbf{w} \leftarrow \mathbf{w} + \Gamma(\nabla F_S(\mathbf{w})) \\
&5 \qquad \textsc{Update}(\mathbf{w} \parallel p \text{ machines}) \\
&6 \quad \textbf{return } \mathbf{w}
\end{aligned}
$$

Figure 2: Distributed Gradient Training

$m$ points drawn i.i.d., $S_1, \ldots, S_p$. At each iteration, the gradients $\nabla F_{S_k}(\mathbf{w})$ are computed by these $p$ machines in parallel. These separate gradients are then summed up to compute the exact global gradient on a single machine, which also performs the optimization step and updates the weight vector received by all other machines (Figure 2). Chu et al. [4] describe a map-reduce formulation for this computation, where each training epoch consists of one map (compute each $\nabla F_{S_k}(\mathbf{w})$) and one reduce (update $\mathbf{w}$). However, the update method they present is that of Newton-Raphson, which requires the computation of the Hessian. We do not consider such strategies, since Hessian computations are often infeasible for large data sets.

## 2.3 Majority Vote Method

The ensemble methods described in the next two paragraphs are based on mixture weights $\boldsymbol{\mu} \in \mathbb{R}^p$. Let $\Delta_p = \{\boldsymbol{\mu} \in \mathbb{R}^p : \boldsymbol{\mu} \geq 0 \wedge \sum_{k=1}^{p} \mu_k = 1\}$ denote the simplex of $\mathbb{R}^p$ and let $\boldsymbol{\mu} \in \Delta_p$. In the absence of any prior knowledge, $\boldsymbol{\mu}$ is chosen to be the uniform mixture $\boldsymbol{\mu}_0 = (1/p, \ldots, 1/p)$ as in all of our experiments.

Instead of computing the gradient of the global function in parallel, a (weighted) majority vote method can be used. Each machine receives one subsample $S_k$, $k \in [1, p]$, and computes $\mathbf{w}_k = \operatorname{argmin}_{\mathbf{w} \in H} F_{S_k}(\mathbf{w})$ by applying the standard training of Figure 1 to $S_k$. The output $y$ predicted by the majority vote method for an input $x$ is

$$
y = \operatorname*{argmax}_{y \in \mathcal{Y}} \sum_{k=1}^{p} \mu_k \, \mathbf{I}(\operatorname*{argmax}_{y' \in \mathcal{Y}} p_{\mathbf{w}_k}[y'|x] = y), \tag{3}
$$

where $\mathbf{I}$ is an indicator function of the predicate it takes as argument. Alternatively, the conditional class probabilities could be used to take into account the uncertainty of each classifier: $y = \operatorname{argmax}_y \sum_{k=1}^{p} \mu_k \, p_{\mathbf{w}_k}[y|x]$.

## 2.4 Mixture Weight Method

The cost of storing $p$ weight vectors can make the majority vote method unappealing. Instead, a single mixture weight $\mathbf{w}_{\boldsymbol{\mu}}$ can be defined form the weight vectors $\mathbf{w}_k, k \in [1, p]$:

$$
\mathbf{w}_{\boldsymbol{\mu}} = \sum_{k=1}^{p} \mu_k \mathbf{w}_k. \tag{4}
$$

The mixture weight $\mathbf{w}_{\boldsymbol{\mu}}$ can be used directly for classification.

## 2.5 Comparison of CPU and Network Times

This section compares the CPU and network time complexity of the three training methods just described. Table 1 summarizes these results. Here, we denote by $N$ the dimension of $H$. *User CPU* represents the CPU time experienced by the user, *cumulative CPU* the total amount of CPU time for the machines participating in the computation, and *latency* the experienced runtime effects due to network activity. The *cumulative network usage* is the amount of data transferred across the network during a distributed computation.

For a training sample of $pm$ points, both the user and cumulative CPU times are in $O_{\mathsf{cpu}}(TpmN)$ when training on a single machine (Figure 1) since at each of the $T$ iterations, the gradient computation must iterate over all $pm$ training points and update all the components of $\mathbf{w}$.

| | Training<br>User CPU + Latency | Training<br>Cum. CPU | Training<br>Cum. Network | Prediction<br>User CPU |
|---|---|---|---|---|
| Single Machine | $O_{\mathsf{cpu}}(pmNT)$ | $O_{\mathsf{cpu}}(pmNT)$ | N/A | $O_{\mathsf{cpu}}(N)$ |
| Distributed Gradient | $O_{\mathsf{cpu}}(mNT) + O_{\mathsf{lat}}(NT)$ | $O_{\mathsf{cpu}}(pmNT)$ | $O_{\mathsf{net}}(pNT)$ | $O_{\mathsf{cpu}}(N)$ |
| Majority Vote | $O_{\mathsf{cpu}}(mNT_{\max}) + O_{\mathsf{lat}}(N)$ | $\sum_{k=1}^{p} O_{\mathsf{cpu}}(mNT_k)$ | $O_{\mathsf{net}}(pN)$ | $O_{\mathsf{cpu}}(pN)$ |
| Mixture Weight | $O_{\mathsf{cpu}}(mNT_{\max}) + O_{\mathsf{lat}}(N)$ | $\sum_{k=1}^{p} O_{\mathsf{cpu}}(mNT_k)$ | $O_{\mathsf{net}}(pN)$ | $O_{\mathsf{cpu}}(N)$ |

Table 1: Comparison of CPU and network times.

For the distributed gradient method (Section 2.2), the worst-case user CPU of the gradient and parameter update computations (lines 3-4 of Figure 2) is $O_{\mathsf{cpu}}(mN+pN+N)$ since each parallel gradient calculation takes $mN$ to compute the gradient for $m$ instances, $p$ gradients of size $N$ need to be summed, and the parameters updated. We assume here that the time to compute $\Gamma$ is negligible. If we assume that $p \ll m$, then, the user CPU is in $O_{\mathsf{cpu}}(mNT)$. Note that the number of iterations it takes to converge, $T$, is the same as when training on a single machine since the computations are identical.

In terms of network usage, a distributed gradient strategy will incur a cost of $O_{\mathsf{net}}(pNT)$ and a latency proportional to $O_{\mathsf{lat}}(NT)$, since at each iteration $\mathbf{w}$ must be transmitted to each of the $p$ machines (in parallel) and each $\nabla F_{S_k}(\mathbf{w})$ returned back to the master. Network time can be improved through better data partitioning of $S$ when $\mathbf{\Phi}(x,y)$ is sparse. The exact runtime cost of latency is complicated as it depends on factors such as the physical distance between the master and each machine, connectivity, the switch fabric in the network, and CPU costs required to manage messages. For parallelization on massively multi-core machines [4], communication latency might be negligible. However, in large data centers running commodity machines, a more common case, network latency cost can be significant.

The training times are identical for the majority vote and mixture weight techniques. Let $T_k$ be the number of iterations for training the $k$th mixture component $\mathbf{w}_k$ and let $T_{\max} = \max\{T_1, \ldots, T_p\}$. Then, the user CPU usage of training is in $O_{\mathsf{cpu}}(mNT_{\max})$, similar to that of the distributed gradient method. However, in practice, $T_{\max}$ is typically less than $T$ since convergence is often faster with smaller data sets. A crucial advantage of these methods over the distributed gradient method is that their network usage is significantly less than that of the distributed gradient computation. While parameters and gradients are exchanged at each iteration for this method, majority vote and mixture weight techniques only require the final weight vectors to be transferred at the conclusion of training. Thus, the overall network usage is $O_{\mathsf{net}}(pN)$ with a latency in $O_{\mathsf{lat}}(NT)$. The main difference between the majority vote and mixture weight methods is the user CPU (and memory usage) for prediction which is in $O_{\mathsf{cpu}}(pN)$ versus $O_{\mathsf{cpu}}(N)$ for the mixture weight method. Prediction could be distributed over $p$ machines for the majority vote method, but that would incur additional machine and network bandwidth costs.

# 3 Theoretical Analysis

This section presents a theoretical analysis of conditional maxent models, including a study of the convergence of the mixture weight method, the most resource-efficient technique, as suggested in the previous section.

The results we obtain are quite general and include the proof of several fundamental properties of the weight vector $\mathbf{w}$ obtained when training a conditional maxent model. We first prove the stability of $\mathbf{w}$ in response to a change in one of the training points. We then give a convergence bound for $\mathbf{w}$ as a function of the sample size in terms of the norm of the feature space and also show a similar result for the mixture weight $\mathbf{w}_{\boldsymbol{\mu}}$. These results are used to compare the weight vector $\mathbf{w}_{pm}$ obtained by training on a sample of size $pm$ with the mixture weight vector $\mathbf{w}_{\boldsymbol{\mu}}$.

Consider two training samples of size $m$, $S = (z_1, \ldots, z_{m-1}, z_m)$ and $S' = (z_1, \ldots, z_{m-1}, z'_m)$, with elements in $\mathcal{X} \times \mathcal{Y}$, that differ by a single training point, which we arbitrarily set as the last one of each sample: $z_m = (x_m, y_m)$ and $z'_m = (x'_m, y'_m)$. Let $\mathbf{w}$ denote the parameter vector returned by conditional maximum entropy when trained on sample $S$, $\mathbf{w}'$ the vector returned when trained on $S'$, and let $\Delta \mathbf{w}$ denote $\mathbf{w}' - \mathbf{w}$. We shall assume that the feature vectors are bounded, that is there exists $R > 0$ such that for all $(x,y)$ in $\mathcal{X} \times \mathcal{Y}$, $\|\mathbf{\Phi}(x,y)\| \leq R$. Our bounds are derived using

techniques similar to those used by Bousquet and Elisseeff [2], or other authors, e.g., [6], in the analysis of stability. In what follows, for any $\mathbf{w} \in H$ and $z = (x, y) \in \mathcal{X} \times \mathcal{Y}$, we denote by $L_z(\mathbf{w})$ the negative log-likelihood $-\log p_{\mathbf{w}}[y|x]$.

**Theorem 1.** *Let $S'$ and $S$ be two arbitrary samples of size $m$ differing only by one point. Then, the following stability bound holds for the weight vector returned by a conditional maxent model:*

$$\|\Delta \mathbf{w}\| \leq \frac{2R}{\lambda m}. \tag{5}$$

*Proof.* We denote by $B_F$ the Bregman divergence associated to a convex and differentiable function $F$ defined for all $\mathbf{u}, \mathbf{u}'$ by: $B_F(\mathbf{u}'\|\mathbf{u}) = F(\mathbf{u}') - F(\mathbf{u}) - \nabla F(\mathbf{u}) \cdot (\mathbf{u}' - \mathbf{u})$. Let $G_S$ denote the function $\mathbf{u} \mapsto \frac{1}{m}\sum_{i=1}^{m} L_{z_i}(\mathbf{u})$ and $W$ the function $\mathbf{u} \mapsto \lambda\|\mathbf{u}\|^2$. $G_S$ and $W$ are convex and differentiable functions. Since the Bregman divergence is non-negative, $B_{G_S} \geq 0$ and $B_{F_S} = B_W + B_{G_S} \geq B_W$. Similarly, $B_{F_{S'}} \geq B_W$. Thus, the following inequality holds:

$$B_W(\mathbf{w}'\|\mathbf{w}) + B_W(\mathbf{w}\|\mathbf{w}') \leq B_{F_S}(\mathbf{w}'\|\mathbf{w}) + B_{F_{S'}}(\mathbf{w}\|\mathbf{w}'). \tag{6}$$

By the definition of $\mathbf{w}$ and $\mathbf{w}'$ as the minimizers of $F_S$ and $F_{S'}$, $\nabla F_S(\mathbf{w}) = \nabla F_{S'}(\mathbf{w}') = \mathbf{0}$ and

$$
\begin{aligned}
B_{F_S}(\mathbf{w}'\|\mathbf{w}) + B_{F_{S'}}(\mathbf{w}\|\mathbf{w}') &= F_S(\mathbf{w}') - F_S(\mathbf{w}) + F_{S'}(\mathbf{w}) - F_{S'}(\mathbf{w}') \\
&= \frac{1}{m}\Big[\big[L_{z_m}(\mathbf{w}') - L_{z_m}(\mathbf{w})\big] + \big[L_{z'_m}(\mathbf{w}) - L_{z'_m}(\mathbf{w}')\big]\Big] \\
&\leq -\frac{1}{m}\Big[\nabla L_{z_m}(\mathbf{w}') \cdot (\mathbf{w} - \mathbf{w}') + \nabla L_{z'_m}(\mathbf{w}) \cdot (\mathbf{w}' - \mathbf{w})\Big] \\
&= -\frac{1}{m}\big[\nabla L_{z'_m}(\mathbf{w}) - \nabla L_{z_m}(\mathbf{w}')\big] \cdot (\mathbf{w}' - \mathbf{w}),
\end{aligned}
$$

where we used the convexity of $L_{z'_m}$ and $L_{z_m}$. It is not hard to see that $B_W(\mathbf{w}'\|\mathbf{w}) + B_W(\mathbf{w}\|\mathbf{w}') = 2\lambda\|\Delta\mathbf{w}\|^2$. Thus, the application of the Cauchy-Schwarz inequality to the inequality just established yields

$$2\lambda\|\Delta\mathbf{w}\| \leq \frac{1}{m}\|\nabla L_{z_m}(\mathbf{w}') - \nabla L_{z'_m}(\mathbf{w})\| \leq \frac{1}{m}\Big[\|\nabla L_{z_m}(\mathbf{w}')\| + \|\nabla L_{z'_m}(\mathbf{w})\|\Big]. \tag{7}$$

The gradient of $\mathbf{w} \mapsto L_{z_m}(\mathbf{w}) = \log \sum_{y \in Y} e^{\mathbf{w} \cdot \mathbf{\Phi}(x_m, y)} - \mathbf{w} \cdot \mathbf{\Phi}(x_m, y_m)$ is given by

$$\nabla L_{z_m}(\mathbf{w}) = \frac{\sum_{y \in Y} e^{\mathbf{w} \cdot \mathbf{\Phi}(x_m, y)}\mathbf{\Phi}(x_m, y)}{\sum_{y' \in Y} e^{\mathbf{w} \cdot \mathbf{\Phi}(x_m, y')}} - \mathbf{\Phi}(x_m, y_m) = \mathop{\mathrm{E}}_{y \sim p_{\mathbf{w}}[\cdot|x_m]}\big[\mathbf{\Phi}(x_m, y) - \mathbf{\Phi}(x_m, y_m)\big].$$

Thus, we obtain $\|\nabla L_{z_m}(\mathbf{w}')\| \leq \mathrm{E}_{y \sim p_{\mathbf{w}'}[\cdot|x_m]}\big[\|\mathbf{\Phi}(x_m, y) - \mathbf{\Phi}(x_m, y_m)\|\big] \leq 2R$ and similarly $\|\nabla L_{z'_m}(\mathbf{w})\| \leq 2R$, which leads to the statement of the theorem. $\qquad\square$

Let $D$ denote the distribution according to which training and test points are drawn and let $F^\star$ be the objective function associated to the optimization defined with respect to the true log loss:

$$F^\star(\mathbf{w}) = \operatorname*{argmin}_{\mathbf{w} \in H} \lambda\|\mathbf{w}\|^2 + \mathop{\mathrm{E}}_{z \sim D}\big[L_z(\mathbf{w})\big]. \tag{8}$$

$F^\star$ is a convex function since $\mathrm{E}_D[L_z]$ is convex. Let the solution of this optimization be denoted by $\mathbf{w}^\star = \operatorname{argmin}_{\mathbf{w} \in H} F^\star(\mathbf{w})$.

**Theorem 2.** *Let $\mathbf{w} \in H$ be the weight vector returned by conditional maximum entropy when trained on a sample $S$ of size $m$. Then, for any $\delta > 0$, with probability at least $1 - \delta$, the following inequality holds:*

$$\|\mathbf{w} - \mathbf{w}^\star\| \leq \frac{R}{\lambda\sqrt{m/2}}\big(1 + \sqrt{\log 1/\delta}\big). \tag{9}$$

*Proof.* Let $S$ and $S'$ be as before samples of size $m$ differing by a single point. To derive this bound, we apply McDiarmid's inequality [17] to $\Psi(S) = \|\mathbf{w} - \mathbf{w}^\star\|$. By the triangle inequality and Theorem 1, the following Lipschitz property holds:

$$|\Psi(S') - \Psi(S)| = \big|\|\mathbf{w}' - \mathbf{w}^\star\| - \|\mathbf{w} - \mathbf{w}^\star\|\big| \leq \|\mathbf{w}' - \mathbf{w}\| \leq \frac{2R}{\lambda m}. \tag{10}$$

Thus, by McDiarmid's inequality, $\Pr[\Psi - \mathrm{E}[\Psi] \geq \epsilon] \leq \exp\left(\frac{-2\epsilon^2 m}{4R^2/\lambda^2}\right)$. The following bound can be shown for the expectation of $\Psi$ (see longer version of this paper): $\mathrm{E}[\Psi] \leq \frac{2R}{\lambda\sqrt{2m}}$. Using this bound and setting the right-hand side of McDiarmid's inequality to $\delta$ show that the following holds

$$\Psi \leq \mathrm{E}[\Psi] + \frac{2R}{\lambda}\sqrt{\frac{\log\frac{1}{\delta}}{2m}} \leq \frac{2R}{\lambda\sqrt{2m}}\left(1 + \sqrt{\log 1/\delta}\right), \tag{11}$$

with probability at least $1-\delta$. $\qquad\square$

Note that, remarkably, the bound of Theorem 2 does not depend on the dimension of the feature space but only on the radius $R$ of the sphere containing the feature vectors.

Consider now a sample $S = (S_1, \ldots, S_p)$ of $pm$ points formed by $p$ subsamples of $m$ points drawn i.i.d. and let $\mathbf{w}_{\boldsymbol{\mu}}$ denote the $\boldsymbol{\mu}$-mixture weight as defined in Section 2.4. The following theorem gives a learning bound for $\mathbf{w}_{\boldsymbol{\mu}}$.

**Theorem 3.** *For any $\boldsymbol{\mu} \in \Delta_p$, let $\mathbf{w}_{\boldsymbol{\mu}} \in H$ denote the mixture weight vector obtained from a sample of size $pm$ by combining the $p$ weight vectors $\mathbf{w}_k$, $k \in [1, p]$, each returned by conditional maximum entropy when trained on the sample $S_k$ of size $m$. Then, for any $\delta > 0$, with probability at least $1 - \delta$, the following inequality holds:*

$$\|\mathbf{w}_{\boldsymbol{\mu}} - \mathbf{w}^\star\| \leq \mathrm{E}\left[\|\mathbf{w}_{\boldsymbol{\mu}} - \mathbf{w}^\star\|\right] + \frac{R\|\boldsymbol{\mu}\|}{\lambda\sqrt{m/2}}\sqrt{\log 1/\delta}. \tag{12}$$

*For the uniform mixture $\boldsymbol{\mu}_0 = (1/p, \ldots, 1/p)$, the bound becomes*

$$\|\mathbf{w}_{\boldsymbol{\mu}} - \mathbf{w}^\star\| \leq \mathrm{E}\left[\|\mathbf{w}_{\boldsymbol{\mu}} - \mathbf{w}^\star\|\right] + \frac{R}{\lambda\sqrt{pm/2}}\sqrt{\log 1/\delta}. \tag{13}$$

*Proof.* The result follows by application of McDiarmid's inequality to $\Upsilon(S) = \|\mathbf{w}_{\boldsymbol{\mu}} - \mathbf{w}^\star\|$. Let $S' = (S'_1, \ldots, S'_p)$ denote a sample differing from $S$ by one point, say in subsample $S_k$. Let $\mathbf{w}'_k$ denote the weight vector obtained by training on subsample $S'_k$ and $\mathbf{w}'_{\boldsymbol{\mu}}$ the mixture weight vector associated to $S'$. Then, by the triangle inequality and the stability bound of Theorem 1, the following holds:

$$|\Upsilon(S') - \Upsilon(S)| = \left|\|\mathbf{w}'_{\boldsymbol{\mu}} - \mathbf{w}^\star\| - \|\mathbf{w}_{\boldsymbol{\mu}} - \mathbf{w}^\star\|\right| \leq \|\mathbf{w}'_{\boldsymbol{\mu}} - \mathbf{w}_{\boldsymbol{\mu}}\| = \mu_k\|\mathbf{w}'_k - \mathbf{w}_k\| \leq \frac{2\mu_k R}{\lambda m}.$$

Thus, by McDiarmid's inequality,

$$\Pr[\Upsilon(S) - \mathrm{E}[\Upsilon(S)] \geq \epsilon] \leq \exp\left(\frac{-2\epsilon^2}{\sum_{k=1}^p m\left(\frac{2\mu_k R}{\lambda m}\right)^2}\right) = \exp\left(\frac{-2\lambda^2 m\epsilon^2}{4R^2\|\boldsymbol{\mu}\|^2}\right), \tag{14}$$

which proves the first statement and the uniform mixture case since $\|\boldsymbol{\mu}_0\| = 1/\sqrt{p}$. $\qquad\square$

Theorems 2 and 3 help us compare the mixture weight $\mathbf{w}_{pm}$ obtained by training on a sample of size $pm$ versus the mixture weight vector $\mathbf{w}_{\boldsymbol{\mu}_0}$. The regularization parameter $\lambda$ is a function of the sample size. To simplify the analysis, we shall assume that $\lambda = O(1/m^{1/4})$ for a sample of size $m$. A similar discussion holds for other comparable asymptotic behaviors. By Theorem 2, $\|\mathbf{w}_{pm} - \mathbf{w}^\star\|$ converges to zero in $O(1/(\lambda\sqrt{pm})) = O(1/(pm)^{1/4})$, since $\lambda = O(1/(pm)^{1/4})$ in that case. But, by Theorem 3, the slack term bounding $\|\mathbf{w}_{\boldsymbol{\mu}_0} - \mathbf{w}^\star\|$ converges to zero at the faster rate $O(1/(\lambda\sqrt{pm})) = O(1/p^{1/2}m^{1/4})$, since here $\lambda = O(1/m^{1/4})$. The expectation term appearing in the bound on $\|\mathbf{w}_{\boldsymbol{\mu}_0} - \mathbf{w}^\star\|$, $\mathrm{E}[\|\mathbf{w}_{\boldsymbol{\mu}_0} - \mathbf{w}^\star\|]$, does not benefit from the same convergence rate however. $\mathrm{E}[\|\mathbf{w}_{\boldsymbol{\mu}_0} - \mathbf{w}^\star\|]$ converges always as fast as the expectation $\mathrm{E}[\|\mathbf{w}_m - \mathbf{w}^\star\|]$ for a weight vector $\mathbf{w}_m$ obtained by training on a sample of size $m$ since, by the triangle inequality, the following holds:

$$\mathrm{E}[\|\mathbf{w}_{\boldsymbol{\mu}} - \mathbf{w}^\star\|] = \mathrm{E}[\|\frac{1}{p}\sum_{k=1}^p(\mathbf{w}_k - \mathbf{w}^\star)\|] \leq \frac{1}{p}\sum_{k=1}^p \mathrm{E}[\|\mathbf{w}_k - \mathbf{w}^\star\|] = \mathrm{E}[\|\mathbf{w}_1 - \mathbf{w}^\star\|]. \tag{15}$$

By the proof of Theorem 2, $\mathrm{E}[\|\mathbf{w}_1 - \mathbf{w}^\star\|] \leq R/(\lambda\sqrt{m/2}) = O(1/(\lambda\sqrt{m}))$, thus $\mathrm{E}[\|\mathbf{w}_{\boldsymbol{\mu}} - \mathbf{w}^\star\|] \leq O(1/m^{1/4})$. In summary, $\mathbf{w}_{\boldsymbol{\mu}_0}$ always converges significantly faster than $\mathbf{w}_m$. The convergence bound for $\mathbf{w}_{\boldsymbol{\mu}_0}$ contains two terms, one somewhat more favorable, one somewhat less than its counterpart term in the bound for $\mathbf{w}_{pm}$.

|                        | $pm$    | $|\mathcal{Y}|$ | $|\mathcal{X}|$ | sparsity | $p$  |
|------------------------|---------|------|-------|----------|------|
| English POS [16]       | 1 M     | 24   | 500 K | 0.001    | 10   |
| Sentiment              | 9 M     | 3    | 500 K | 0.001    | 10   |
| RCV1-v2 [14]           | 26 M    | 103  | 10 K  | 0.08     | 10   |
| Speech                 | 50 M    | 129  | 39    | 1.0      | 499  |
| Deja News Archive      | 306 M   | 8    | 50 K  | 0.002    | 200  |
| Deja News Archive 250K | 306 M   | 8    | 250 K | 0.0004   | 200  |
| Gigaword [10]          | 1,000 M | 96   | 10 K  | 0.001    | 1000 |

Table 2: Description of data sets. The column named sparsity reports the frequency of non-zero feature values for each data set.

## 4 Experiments

We ran a number of experiments on data sets ranging in size from 1M to 1B labeled instances (see Table 2) to compare the three distributed training methods described in Section 2. Our experiments were carried out using a large cluster of commodity machines with a local shared disk space and a high rate of connectivity between each machine and between machines and disk. Thus, while the processes did not run on one multi-core supercomputer, the network latency between machines was minimized.

We report accuracy, wall clock, cumulative CPU usage, and cumulative network usage for all of our experiments. Wall clock measures the combined effects of the user CPU and latency costs (column 1 of Table 1), and includes the total time for training, including all summations. Network usage measures the amount of data transferred across the network. Due to the set-up of our cluster, this includes both machine-to-machine traffic and machine-to-disk traffic. The resource estimates were calculated by point-sampling and integrating over the sampling time. For all three methods, we used the same base implementation of conditional maximum entropy, modified only in whether or not the gradient was computed in a distributed fashion.

Our first set of experiments were carried out with "medium" scale data sets containing 1M-300M instances. These included: `English part-of-speech` tagging, generated from the Penn Treebank [16] using the first character of each part-of-speech tag as output, sections 2-21 for training, section 23 for testing and a feature representation based on the identity, affixes, and orthography of the input word and the words in a window of size two; `Sentiment analysis`, generated from a set of online product, service, and merchant reviews with a three-label output (positive, negative, neutral), with a bag of words feature representation; `RCV1-v2` as described by [14], where documents having multiple labels were included multiple times, once for each label; `Acoustic Speech Data`, a 39-dimensional input consisting of 13 PLP coefficients, plus their first and second derivatives, and 129 outputs (43 phones × 3 acoustic states); and the `Deja News Archive`, a text topic classification problem generated from a collection of Usenet discussion forums from the years 1995-2000. For all text experiments, we used random feature mixing [9, 20] to control the size of the feature space.

The results reported in Table 3 show that the accuracy of the mixture weight method consistently matches or exceeds that of the majority vote method. As expected, the resource costs here are similar, with slight differences due to the point-sampling methods and the overhead associated with storing $p$ models in memory and writing them to disk. For some data sets, we could not report majority vote results as all models could not fit into memory on a single machine.

The comparison shows that in some cases the mixture weight method takes longer and achieves somewhat better performance than the distributed gradient method while for other data sets it terminates faster, at a slight loss in accuracy. These differences may be due to the performance of the optimization with respect to the regularization parameter $\lambda$. However, the results clearly demonstrate that the mixture weight method achieves comparable accuracies at a much decreased cost in network bandwidth – upwards of 1000x. Depending on the cost model assessed for the underlying network and CPU resources, this may make mixture weight a significantly more appealing strategy. In particular, if network usage leads to significant increases in latency, unlike our current experimental set-up of high rates of connectivity, then the mixture weight method could be substantially faster to train. The outlier appears to be the acoustic speech data, where both mixture weight and distributed gradient have comparable network usage, 158GB and 200GB, respectively. However, the bulk of this comes from the fact that the data set itself is 157GB in size, which makes the network

| | Training Method | Accuracy | Wall Clock | Cumulative CPU | Network Usage |
|---|---|---|---|---|---|
| English POS (m=100k,p=10) | Distributed Gradient | 97.60% | 17.5 m | 11.0 h | 652 GB |
| | Majority Vote | 96.80% | 12.5 m | 18.5 h | 0.686 GB |
| | Mixture Weight | 96.80% | 5 m | 11.5 h | 0.015 GB |
| Sentiment (m=900k,p=10) | Distributed Gradient | 81.18% | 104 m | 123 h | 367 GB |
| | Majority Vote | 81.25% | 131 m | 168 h | 3 GB |
| | Mixture Weight | 81.30% | 110 m | 163 h | 9 GB |
| RCV1-v2 (m=2.6M,p=10) | Distributed Gradient | 27.03% | 48 m | 407 h | 479 GB |
| | Majority Vote | 26.89% | 54 m | 474 h | 3 GB |
| | Mixture Weight | 27.15% | 56 m | 473 h | 0.108 GB |
| Speech (m=100k,p=499) | Distributed Gradient | 34.95% | 160 m | 511 h | 200 GB |
| | Mixture Weight | 34.99% | 130 m | 534 h | 158 GB |
| Deja (m=1.5M,p=200) | Distributed Gradient | 64.74% | 327 m | 733 h | 5,283 GB |
| | Mixture Weight | 65.46% | 316 m | 707 h | 48 GB |
| Deja 250K (m=1.5M,p=200) | Distributed Gradient | 67.03% | 340 m | 698 h | 17,428 GB |
| | Mixture Weight | 66.86% | 300 m | 710 h | 65 GB |
| Gigaword (m=1M,p=1k) | Distributed Gradient | 51.16% | 240 m | 18,598 h | 13,000 GB |
| | Mixture Weight | 50.12% | 215 m | 17,998 h | 21 GB |

Table 3: Accuracy and resource costs for distributed training strategies.

usage closer to 1GB for the mixture weight and 40GB for distributed gradient method when we discard machine-to-disk traffic.

For the largest experiment, we examined the task of predicting the next character in a sequence of text [19], which has implications for many natural language processing tasks. As a training and evaluation corpus we used the English Gigaword corpus [10] and used the full ASCII output space of that corpus of around 100 output classes (uppercase and lowercase alphabet characters variants, digits, punctuation, and whitespace). For each character $s$, we designed a set of observed features based on substrings from $s_{-1}$, the previous character, to $s_{-10}$, 9 previous characters, and hashed each into a 10k-dimensional space in an effort to improve speed. Since there were around 100 output classes, this led to roughly 1M parameters. We then sub-sampled 1B characters from the corpus as well as 10k testing characters and established a training set of 1000 subsets, of 1M instances each. For the experiments described above, the regularization parameter $\lambda$ was kept fixed across the different methods. Here, we decreased the parameter $\lambda$ for the distributed gradient method since less regularization was needed when more data was available, and since there were three orders of magnitude difference between the training size for each independent model and the distributed gradient. We compared only the distributed gradient and mixture weight methods since the majority vote method exceeded memory capacity. On this data set, the network usage is on a different scale than most of the previous experiments, though comparable to Deja 250, with the distributed gradient method transferring 13TB across the network. Overall, the mixture weight method consumes less resources: less bandwidth and less time (both wall clock and CPU). With respect to accuracy, the mixture weight method does only slightly worse than the distributed gradient method. The individual models in the mixture weight method ranged between 49.73% to 50.26%, with a mean accuracy of 50.07%, so a mixture weight model improves slightly over a random subsample models and decreases the overall variance.

# 5 Conclusion

Our analysis and experiments give significant support for the mixture weight method for training very large-scale conditional maximum entropy models with $L_2$ regularization. Empirical results suggest that this method achieves similar or better accuracies while reducing network usage by about three orders of magnitude and modestly reducing the wall clock time, typically by about 15% or more. In distributed environments without a high rate of connectivity, the decreased network usage of the mixture weight method should lead to substantial gains in wall clock as well.

### Acknowledgments

We thank Yishay Mansour for his comments on an earlier version of this paper.

## Footnotes

[1]A batch parallel estimation technique for maxent models based on their connection with AdaBoost is also described by [5]. This algorithm is quite different from the distributed gradient computation method, but, as for that method, it requires a substantial amount of network resources, since updates need to be transferred to the master at every iteration.

# References

[1] A. Berger, V. Della Pietra, and S. Della Pietra. A maximum entropy approach to natural language processing. *Computational Linguistics*, 22(1):39–71, 1996.

[2] O. Bousquet and A. Elisseeff. Stability and generalization. *Journal of Machine Learning Research*, 2:499–526, 2002.

[3] S. F. Chen and R. Rosenfeld. A survey of smoothing techniques for ME models. *IEEE Transactions on Speech and Audio Processing*, 8(1):37–50, 2000.

[4] C. Chu, S. Kim, Y. Lin, Y. Yu, G. Bradski, A. Ng, and K. Olukotun. Map-Reduce for machine learning on multicore. In *Advances in Neural Information Processing Systems*, 2007.

[5] M. Collins, R. Schapire, and Y. Singer. Logistic regression, AdaBoost and Bregman distances. *Machine Learning*, 48, 2002.

[6] C. Cortes, M. Mohri, M. Riley, and A. Rostamizadeh. Sample selection bias correction theory. In *Proceedings of ALT 2008*, volume 5254 of *LNCS*, pages 38–53. Springer, 2008.

[7] J. Darroch and D. Ratcliff. Generalized iterative scaling for log-linear models. *The Annals of Mathematical Statistics*, pages 1470–1480, 1972.

[8] S. Della Pietra, V. Della Pietra, J. Lafferty, R. Technol, and S. Brook. Inducing features of random fields. *IEEE transactions on pattern analysis and machine intelligence*, 19(4):380–393, 1997.

[9] K. Ganchev and M. Dredze. Small statistical models by random feature mixing. In *Workshop on Mobile Language Processing, ACL*, 2008.

[10] D. Graff, J. Kong, K. Chen, and K. Maeda. English gigaword third edition, linguistic data consortium, philadelphia, 2007.

[11] E. T. Jaynes. Information theory and statistical mechanics. *Physical Review*, 106(4):620630, 1957.

[12] J. Jeon and R. Manmatha. Using maximum entropy for automatic image annotation. In *International Conference on Image and Video Retrieval*, 2004.

[13] G. Lebanon and J. Lafferty. Boosting and maximum likelihood for exponential models. In *Advances in Neural Information Processing Systems*, pages 447–454, 2001.

[14] D. Lewis, Y. Yang, T. Rose, and F. Li. RCV1: A new benchmark collection for text categorization research. *Journal of Machine Learning Research*, 5:361–397, 2004.

[15] R. Malouf. A comparison of algorithms for maximum entropy parameter estimation. In *International Conference on Computational Linguistics (COLING)*, 2002.

[16] M. Marcus, M. Marcinkiewicz, and B. Santorini. Building a large annotated corpus of English: The Penn Treebank. *Computational linguistics*, 19(2):313–330, 1993.

[17] C. McDiarmid. On the method of bounded differences. In *Surveys in Combinatorics*, pages 148–188. Cambridge University Press, Cambridge, 1989.

[18] J. Nocedal and S. Wright. *Numerical optimization*. Springer, 1999.

[19] C. E. Shannon. Prediction and entropy of printed English. *Bell Systems Technical Journal*, 30:50–64, 1951.

[20] K. Weinberger, A. Dasgupta, J. Langford, A. Smola, and J. Attenberg. Feature hashing for large scale multitask learning. In *International Conference on Machine Learning*, 2009.

[21] T. Zhang. Solving large scale linear prediction problems using stochastic gradient descent algorithms. In *International Conference on Machine Learning*, 2004.

